# Weight Space Probability Densities in Stochastic Learning: II. Transients and Basin Hopping Times

**Genevieve B. Orr and Todd K. Leen**
Department of Computer Science and Engineering
Oregon Graduate Institute of Science & Technology
19600 N.W. von Neumann Drive
Beaverton, OR 97006-1999

## Abstract

In stochastic learning, weights are random variables whose time evolution is governed by a Markov process. At each time-step, $n$, the weights can be described by a probability density function $P(\omega, n)$. We summarize the theory of the time evolution of $P$, and give graphical examples of the time evolution that contrast the behavior of stochastic learning with true gradient descent (batch learning). Finally, we use the formalism to obtain predictions of the time required for noise-induced hopping between basins of different optima. We compare the theoretical predictions with simulations of large ensembles of networks for simple problems in supervised and unsupervised learning.

## 1 Weight-Space Probability Densities

Despite the recent application of convergence theorems from stochastic approximation theory to neural network learning (Oja 1982, White 1989) there remain outstanding questions about the search dynamics in stochastic learning. For example, the convergence theorems do not tell us to which of several optima the algorithm

is likely to converge[1]. Also, while it is widely recognized that the intrinsic noise in the weight update can move the system out of sub-optimal local minima (for a graphical example, see Darken and Moody 1991), there have been no theoretical predictions of the time required to escape from local optima, or of its dependence on learning rates.

In order to more fully understand the dynamics of stochastic search, we study the weight-space probability density and its time evolution. In this paper we summarize a theoretical framework that describes this time evolution. We graphically portray the motion of the density for examples that contrast stochastic and batch learning. Finally we use the theory to predict the statistical distribution of times required for escape from local optima. We compare the theoretical results with simulations for simple examples in supervised and unsupervised learning.

## 2   Stochastic Learning and Noisy Maps

### 2.1   Motion of the Probability Density

We consider stochastic learning algorithms of the form

$$\omega(n+1) \ = \ \omega(n) \ + \ \mu H[\omega(n), x(n)] \tag{1}$$

where $\omega(n) \in \mathbb{R}^m$ is the weight, $x(n)$ is the data exemplar input to the algorithm at time-step $n$, $\mu$ is the learning rate, and $H[\cdots] \in \mathbb{R}^m$ is the weight update function. The exemplars $x(n)$ can be either inputs or, in the case of supervised learning, input/target pairs. We assume that the $x(n)$ are i.i.d. with density $\rho(x)$. Angled brackets $\langle \ldots \rangle_x$ denote averaging over this density. In what follows, the learning rate will be held constant.

The learning algorithm (1) is a noisy map on $\omega$. The weights are thus random variables described by the probability density function $P(\omega, n)$. The time evolution of this density is given by the Kolmogorov equation

$$P(\omega, n+1) \ = \ \int d\omega' \, P(\omega', n) \, W(\omega' \to \omega) \tag{2}$$

where the single time-step transition probability is given by (Leen and Orr 1992, Leen and Moody 1993)

$$W(\omega' \to \omega) \ = \ \langle \, \delta(\,\omega - \omega' - \mu H[\omega', x]\,) \, \rangle_x \tag{3}$$

and $\delta(\cdots)$ is the Dirac delta function.

The Kolmogorov equation can be recast as a differential-difference equation by expanding the transition probability (3) as a power series in $\mu$. This gives a Kramers-Moyal expansion (Leen and Orr 1992, Leen and Moody 1993)

$$P(\omega, n+1) \; - \; P(\omega, n) \; =$$

$$\sum_{i=1}^{\infty} \frac{(-\mu)^i}{i!} \sum_{j_1, \dots, j_i = 1}^{m} \frac{\partial^i}{\partial \omega_{j_1} \partial \omega_{j_2} \dots \partial \omega_{j_i}} \left( \langle H_{j_1} H_{j_2} \dots H_{j_i} \rangle_x \, P(\omega, n) \right), \qquad (4)$$

where $\omega_{j_\alpha}$ and $H_{j_\alpha}$ are the $j_\alpha^{th}$ component of weight, and weight update, respectively.

Truncating (4) to second order in $\mu$ leaves a Fokker-Planck equation[2] that is valid for small $|\mu H|$. The drift coefficient $\langle H \rangle_x$ is simply the average update. It is important to note that the diffusion coefficients, $\langle H_{j_\alpha} H_{j_\beta} \rangle_x$, can be *strongly* dependent on location in the weight-space. This spatial dependence influences both equilibria and transient phenomena. In section 3.1 we will use both the Kolmogorov equation (2), and the Fokker-Planck equation to track the time evolution of network ensemble densities.

## 2.2   First Passage Times

Our discussion of basin hopping will use the notion of the first passage time (Gardiner, 1990); the time required for a network initialized at $\omega_0$ to first pass into an $\epsilon$-neighborhood $D$ of a global or local optimum $\omega_*$ (see Figure 1). The first passage time is a random variable. Its distribution function $\mathcal{P}(n; \omega_0)$ is the probability that a network initialized at $\omega_0$ makes its first passage into $D$ at the $n^{th}$ iteration of the learning rule.

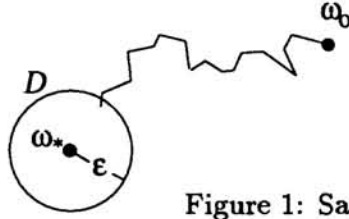

Figure 1: Sample search path.

To arrive at an expression for $\mathcal{P}(n; \omega_0)$, we first examine the probability of passing from the initial weight $\omega_0$ to the weight $\omega$ after $n$ iterations. This probability can be expressed as

$$P(\omega, n \mid \omega_0, 0) \; = \; \int d\omega' \, P(\omega, n \mid \omega', 1) \, W(\omega_0 \to \omega'). \qquad (5)$$

Substituting the single time-step transition probability (3) into the above expression, integrating over $\omega'$, and making use of the time-shift invariance of the system[3] we find

$$P(\omega, n \mid \omega_0, 0) = \langle P(\omega, n-1 \mid \omega_0 + \mu H(\omega_0, x), 0) \rangle_x . \qquad (6)$$

Next, let $G(n; \omega_0)$ denote the probability that a network initialized at $\omega_0$ has *not* passed into the region $D$ by the $n^{th}$ iteration. We obtain $G(n; \omega_0)$ by integrating $P(\omega, n \mid \omega_0, 0)$ over weights $\omega$ *not in $D$*;

$$G(n; \omega_0) \; = \; \int_{D^c} d\omega \, P(\omega, n \mid \omega_0, 0) \qquad (7)$$

where $D^c$ is the complement of $D$. Substituting equation (6) into (7) and integrating over $\omega$ we obtain the recursion

$$G(n; \omega_0) = \langle\, G(n-1; \omega_0 + \mu H[\omega_0, x]) \,\rangle_x \; . \tag{8}$$

Before any learning takes place, none of the networks in the ensemble have entered $D$. Thus the initial condition for $G$ is

$$G(0; \omega_0) = 1 , \quad \omega_0 \in D^c \; . \tag{9}$$

Networks that have entered $D$ are removed from the ensemble (i.e. $\partial D$ is an absorbing boundary). Thus $G$ satisfies the boundary condition

$$G(n; \omega_0) = 0 , \quad \omega_0 \in D \; . \tag{10}$$

Finally, the probability that the network has *not* passed into the region $D$ on or before iteration $n - 1$ minus the probability the network has *not* passed into $D$ on or before iteration $n$ is simply the probability that the network *has* passed into $D$ *exactly at iteration* $n$. This is just the probability for first passage into $D$ at time-step $n$. Thus

$$\mathcal{P}(n; \omega_0) = G(n-1; \omega_0) - G(n; \omega_0) \; . \tag{11}$$

Finally the recursion (8) for $G$ can be expanded in a power series in $\mu$ to obtain the *backward* Kramers-Moyal equation

$$G(n; \omega) - G(n-1; \omega) =$$

$$\sum_{i=1}^{\infty} \frac{\mu^i}{i!} \sum_{j_1,\dots,j_i=1}^{m} \langle H_{j_1} H_{j_2} \dots H_{j_i} \rangle_x \; \frac{\partial^i}{\partial \omega_{j_1} \partial \omega_{j_2} \dots \partial \omega_{j_i}} \, G(n-1; \omega) \; . \tag{12}$$

Truncation to second order in $\mu$ results in the *backward* Fokker-Planck equation. In section 3.2 we will use both the full recursion (8) and the Fokker-Planck approximation to (12) to predict basin hopping times in stochastic learning.

## 3     Backpropagation and Competitive Nets

We apply the above formalism to study the time evolution of the probability density for simple backpropagation and competitive learning problems. We give graphical examples of the time evolution of the weight space density, and calculate times for passage from local to global optima.

### 3.1     Densities for the XOR Problem

Feed-forward networks trained to solve the XOR problem provide an example of supervised learning with well-characterized local optima (Lisboa and Perantonis, 1991). We use a 2-input, 2-hidden, 1-output network (9 weights) trained by stochastic gradient descent on the cross-entropy error function in Lisboa and Perantonis (1991). For computational tractability, we reduce the state space dimension by

constraining the search to one- or two-dimensional subspaces of the weight space. To provide global optima at *finite* weight values, the output targets are set to $\delta$ and $1 - \delta$, with $\delta \ll 1$.

Figure 2a shows the cost function evaluated along a line in the weight space. This line, parameterized by $v$, is chosen to pass through a global optimum at $v = 0$, and a local optimum at $v = 1.0$ . In this one-dimensional slice, another local optimum occurs at $v = 1.24$ . Figure 2b shows the evolution of $P(v, n)$ obtained by numerical integration of the Fokker-Planck equation. Figure 2c shows the evolution of $P(v, n)$ estimated by simulation of 10,000 networks, each receiving a different random sequence of the four input/target patterns. Initially the density is peaked up about the local optimum at $v = 1.24$. At intermediate times, there is a spike of density at the local optimum at $v = 1.0$. This spike is narrow since the diffusion coefficient is small there. At late times the density collects at the global optimum. We note that for the learning rate used here, the local optimum at $v = 1.24$ is asymptotically stable under true gradient descent, and no escape would occur.

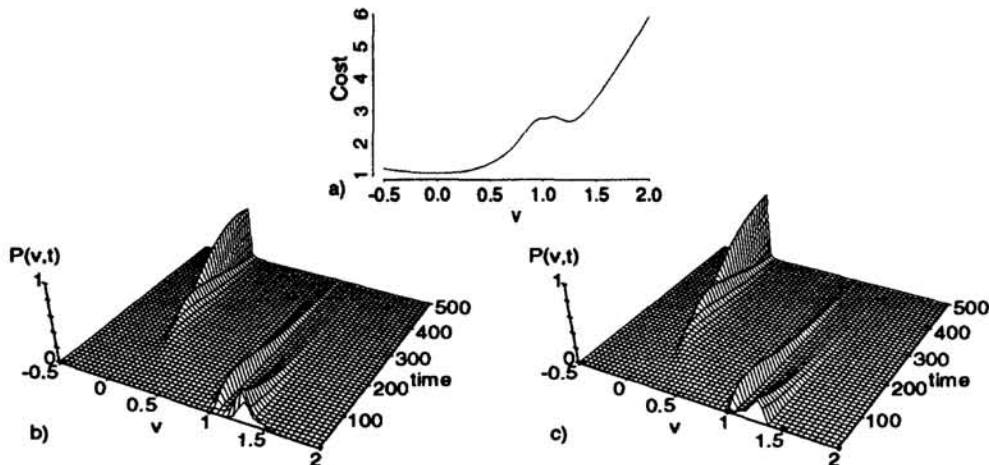

Figure 2: a) XOR cost function. b) Predicted density. c) Simulated density.

Figure 3 shows a series of snapshots of the density superimposed on the cost function for a 2-D slice through the XOR weight space. The first frame shows the weight evolution under true gradient descent. The weights are initialized at the upper right-hand corner of the frame, travel down the gradient and settle into a *local* optimum. The remaining frames show the evolution of the density calculated by direct integration of the Kolmogorov equation (2). Here one sees an early spreading of the initial density and the ultimate concentration at the global optimum.

## 3.2   Basin Hopping Times

The above examples graphically illustrate the intuitive notion that the noise inherent in stochastic learning can move the system out of local optima[4] In this section we calculate the statistical distribution of times required to pass between basins.

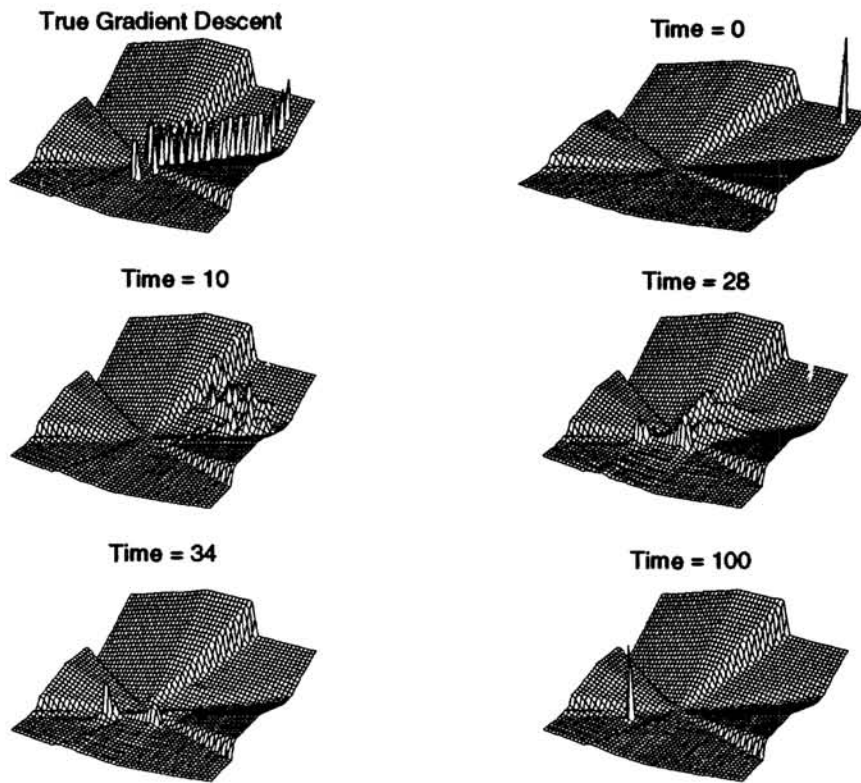

Figure 3: Weight evolution for 2-D XOR. The density is superimposed on top of the cost function. The first frame shows density using true gradient descent for all 100 timesteps. The remaining frames show the density for selected timesteps using stochastic descent.

### 3.2.1    Basin Hopping in Back-propagation

For the search direction used in the example of Figure 2, we calculated the distribution of times required for networks initialized at $v = 1.2$ to first pass within $\epsilon = 0.1$ of the global optimum at $v = 0.0$. For this example we numerically integrated the backward Fokker-Planck equation. We verified the theoretical predictions by obtaining first passage times from an ensemble of 10,000 networks initialized at $v = 1.2$. See Figure 4. For this example the agreement is good at the small learning rate ($\mu = 0.025$) used, but degrades for larger $\mu$ as higher order terms in the expansion (12) become significant.

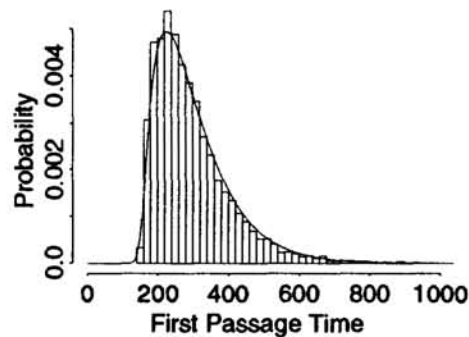

Figure 4: XOR problem. Simulated (histogram) and theoretical (solid line) distributions of first passage times for the cost function of Figure 1a.

When the Fokker-Planck approximation fails, results obtained from the exact expression (8) are in excellent agreement with experimental results. One such example is shown in Figure 5. Similar to Figure 2a, we have chosen a one-dimensional subspace of the XOR weight space (but in a different direction). Here, the Fokker-Planck solution is quite poor because the steepness of the cost function results in large contributions from higher order terms in (12). As one would expect, the exact solution obtained using (8) agrees well with the simulations.

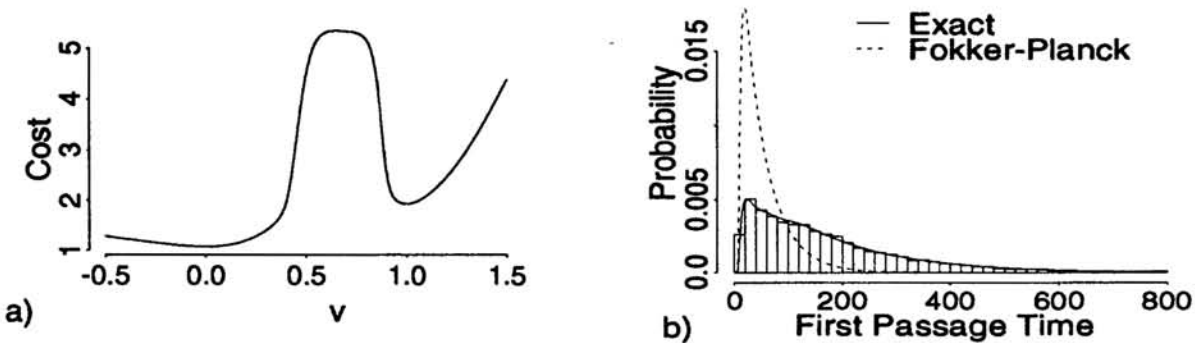

Figure 5: Second 1-D XOR example. a) Cost function. b) Simulated (histogram) and theoretical (lines) distributions of first passage times.

### 3.2.2 Basin Hopping in Competitive Learning

As a final example, we consider competitive learning with two 2-D weight vectors symmetrically placed about the center of a rectangle. Inputs are uniformly distributed in a rectangle of width 1.1 and height 1. This configuration has both global and local optima.

Figure 6a shows a sample path with weights started near the local optimum (crosses) and switching to hover around the global optimum. The measured and predicted (from numerical integration of (8)) distribution of times required to first pass within a distance $\epsilon = 0.1$ of the global optimum are shown in Figure 6b.

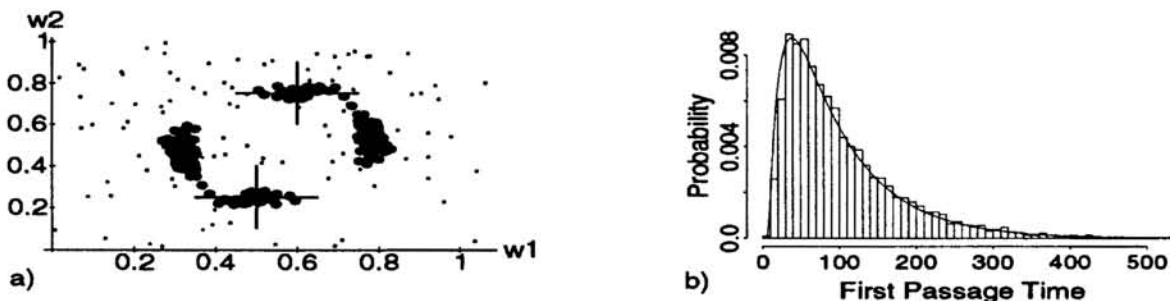

Figure 6: Competitive Learning a) Data (small dots) and sample weight path (large dots). b) First passage times.

## 4   Discussion

The dynamics of the time evolution of the weight space probability density provides a direct handle on the performance of learning algorithms. This paper has focused

on transient phenomena in stochastic learning with constant learning rate. The same theoretical framework can be used to analyze the asymptotic properties of stochastic search with decreasing learning rates, and to analyze equilibrium densities. For a discussion of the latter, see the companion paper in this volume (Leen and Moody 1993).

## Acknowledgements

This work was supported under grants N00014-90-J-1349 and N000-91-J-1482 from the Office of Naval Research.

## Footnotes

[1]However Kushner (1987) has proved convergence to global optima for stochastic approximation algorithms with added Gaussian noise subject to logarithmic annealing schedules.

[2] See (Ritter and Schulten 1988) and (Radons *et al.* 1990) for independent derivations.

[3] With our assumptions of a constant learning rate $\mu$ and stationary sample density $\rho(x)$, the system is time-shift invariant. Mathematically stated, $P(\omega, n \mid \omega', m) = P(\omega, n-1 \mid \omega', m-1)$

[4]The reader should not infer from these examples that stochastic update necessarily converges to global optima. It is straightforward to construct examples for which stochastic learning convergences to local optima with probability one.

## References

E. Oja (1982), A simplified neuron model as a principal component analyzer. *J. Math. Biology*, 15:267–273.

Halbert White (1989), Learning in artificial neural networks: A statistical perspective. *Neural Computation*, 1:425–464.

J.J. Kushner (1987), Asymptotic global behavior for stochastic approximation and diffusions with slowly decreasing noise effects: Global minimization via monte carlo. *SIAM J. Appl. Math.*, 47:169–185.

Christian Darken and John Moody (1991), Note on learning rate schedules for stochastic optimization. In *Advances in Neural Information Processing Systems 3*, San Mateo, CA, Morgan Kaufmann.

Todd K. Leen and Genevieve B. Orr. (1992), Weight-space probability densities and convergence times for stochastic learning. In *International Joint Conference on Neural Networks*, pages IV 158–164. IEEE.

Todd K. Leen and John Moody (1993), Probability Densities in Stochastic Learning: Dynamics and Equilibria. In Giles, C.L., Hanson, S.J., and Cowan, J.D. (eds.), *Advances in Neural Information Processing Systems 5*. San Mateo, CA: Morgan Kaufmann Publishers.

H. Ritter and K. Schulten (1988), Convergence properties of Kohonen's topology conserving maps: Fluctuations, stability and dimension selection, *Biol. Cybern.*, 60, 59-71.

G. Radons, H.G. Schuster and D. Werner (1990), Fokker-Planck description of learning in backpropagation networks, *International Neural Network Conference*, Paris, II 993-996, Kluwer Academic Publishers.

C.W. Gardiner (1990), *Handbook of Stochastic Methods, 2nd Ed.* Springer-Verlag, Berlin.

P. Lisboa and S. Perantonis (1991), Complete solution of the local minima in the XOR problem. *Network: Computation in Neural Systems*, 2:119.